# Learning Multi-level Sparse Representations

**Ferran Diego**     **Fred A. Hamprecht**
Heidelberg Collaboratory for Image Processing (HCI)
Interdisciplinary Center for Scientific Computing (IWR)
University of Heidelberg, Heidelberg 69115, Germany
`{ferran.diego,fred.hamprecht}@iwr.uni-heidelberg.de`

## Abstract

Bilinear approximation of a matrix is a powerful paradigm of unsupervised learning. In some applications, however, there is a natural hierarchy of concepts that ought to be reflected in the unsupervised analysis. For example, in the neurosciences image sequence considered here, there are the semantic concepts of pixel → neuron → assembly that should find their counterpart in the unsupervised analysis. Driven by this concrete problem, we propose a decomposition of the matrix of observations into a product of more than two sparse matrices, with the rank decreasing from lower to higher levels. In contrast to prior work, we allow for both hierarchical and heterarchical relations of lower-level to higher-level concepts. In addition, we *learn* the nature of these relations rather than imposing them. Finally, we describe an optimization scheme that allows to optimize the decomposition over all levels jointly, rather than in a greedy level-by-level fashion.

The proposed bilevel SHMF (sparse heterarchical matrix factorization) is the first formalism that allows to simultaneously interpret a calcium imaging sequence in terms of the constituent neurons, their membership in assemblies, and the time courses of both neurons and assemblies. Experiments show that the proposed model fully recovers the structure from difficult synthetic data designed to imitate the experimental data. More importantly, bilevel SHMF yields plausible interpretations of real-world Calcium imaging data.

## 1 Introduction

This work was stimulated by a concrete problem, namely the decomposition of state-of-the-art $2D + time$ calcium imaging sequences as shown in Fig. 1 into neurons, and assemblies of neurons [20]. Calcium imaging is an increasingly popular tool for unraveling the network structure of local circuits of the brain [11, 6, 7]. Leveraging sparsity constraints seems natural, given that the neural activations are sparse in both space and time. The experimentally achievable optical slice thickness still results in spatial overlap of cells, meaning that each pixel can show intensity from more than one neuron. In addition, it is anticipated that one neuron can be part of more than one assembly. All neurons of an assembly are expected to fire at roughly the same time [20].

A standard sparse decomposition of the set of vectorized images into a dictionary and a set of coefficients would not conform with prior knowledge that we have entities at three levels: the pixels, the neurons, and the assemblies, see Fig. 2. Also, it would not allow to include structured constraints [10] in a meaningful way. As a consequence, we propose a multi-level decomposition (Fig. 3) that

- allows enforcing (structured) sparsity constraints at each level,
- admits both hierarchical or heterarchical relations between levels (Fig. 2),
- can be learned jointly (section 2 and 2.4), and
- yields good results on real-world experimental data (Fig. 2).

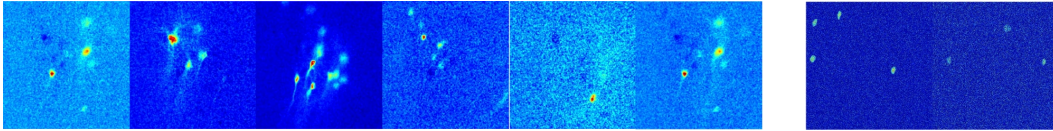

Figure 1: Left: frames from a calcium imaging sequence showing firing neurons that were recorded by an epi-fluorescence microscope. Right: two frames from a synthetic sequence. The underlying biological aim motivating these experiments is to study the role of neuronal assemblies in memory consolidation.

## 1.1 Relation to Previous Work

Most important unsupervised data analysis methods such as PCA, NMF / pLSA, ICA, cluster analysis, sparse coding and others can be written in terms of a bilinear decomposition of, or approximation to, a two-way matrix of raw data [22]. One natural generalization is to perform *multi*linear decompositions of multi-way arrays [4] using methods such as higher-order SVD [1]. This is not the direction pursued here, because the image sequence considered does not have a tensorial structure.

On the other hand, there is a relation to (hierarchical) topic models (e.g. [8]). These do not use structured sparsity constraints, but go beyond our approach in automatically estimating the appropriate number of levels using nonparametric Bayesian models.

Closest to our proposal are four lines of work that we build on: Jenatton *et al.* [10] introduce structured sparsity constraints that we use to find dictionary basis functions representing single neurons. The works [9] and [13] enforce hierarchical (tree-structured) sparsity constraints. These authors find the tree structure using extraneous methods, such as a separate clustering procedure. In contrast, the method proposed here can infer either hierarchical (tree-structured) or heterarchical (directed acyclic graph) relations between entities at different levels. Cichocki and Zdunek [3] proposed a multilayer approach to non-negative matrix factorization. This is a multi-stage procedure which iteratively decomposes the rightmost matrix of the decomposition that was previously found. Similar approaches are explored in [23], [24]. Finally, Rubinstein *et al.* [21] proposed a novel dictionary structure where each basis function in a dictionary is a linear combination of a few elements from a fixed base dictionary. In contrast to these last two methods, we optimize over all factors (including the base dictionary) jointly. Note that our semantics of "bilevel factorization" (section 2.2) are different from the one in [25].

**Notation**. A matrix is a set of columns and rows, respectively, $\mathbf{X} = [\mathbf{x}_{:1}, \ldots, \mathbf{x}_{:n}] = [\mathbf{x}_{1:}; \ldots; \mathbf{x}_{m:}]$. The zero matrix or vector is denoted $\mathbf{0}$, with dimensions inferred from the context. For any vector $\mathbf{x} \in \mathbb{R}^m$, $\|\mathbf{x}\|_\alpha = (\sum_{i=1}^m |x_i|^\alpha)^{1/\alpha}$ is the $l_\alpha$ (quasi-)norm of $\mathbf{x}$, and $\|\cdot\|_F$ is the Frobenius norm.

## 2 Learning a Sparse Heterarchical Structure

### 2.1 Dictionary Learning: Single Level Sparse Matrix Factorization

Let $\mathbf{X} \in \mathbb{R}^{m \times n}$ be a matrix whose $n$ columns represent an $m$-dimensional observation each. The idea of dictionary learning is to find a decomposition $\mathbf{X} \approx \mathbf{D} \left[ \mathbf{U}^0 \right]^T$, see Fig. 3(a). $\mathbf{D}$ is called the dictionary, and its columns hold the basis functions in terms of which the sparse coefficients in $\mathbf{U}^0$ approximate the original observations. The regularization term $\Omega_U$ encourages sparsity of the coefficient matrix. $\Omega_D$ prevents the inflation of dictionary entries to compensate for small coefficients, and induces, if desired, additional structure on the learned basis functions [16]. Interesting theoretical results on support recovery, furthered by an elegantly compact formulation and the ready availability of optimizers [17] have spawned a large number of intriguing and successful applications, *e.g.* image denoising [19] and detection of unusual events [26]. Dictionary learning is a special instance of our framework, involving only a single-level decomposition. In the following we first generalize to two, then to more levels.

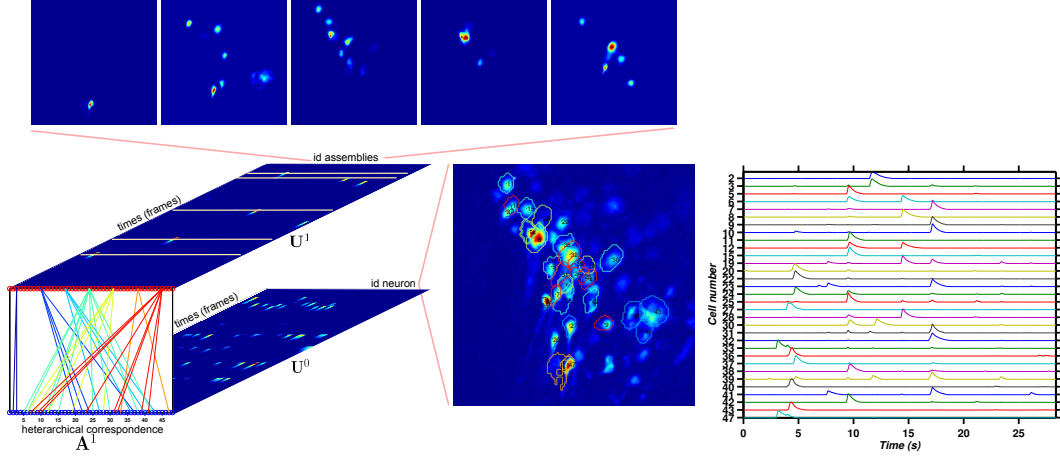

Figure 2: Bottom left: Shown are the temporal activation patterns of individual neurons $\mathbf{U}^0$ (lower level), and assemblies of neurons $\mathbf{U}^1$ (upper level). Neurons $\mathbf{D}$ and assemblies are related by a bipartite graph $\mathbf{A}^1$ the estimation of which is a central goal of this work. The signature of five neuronal assemblies (five columns of $\mathbf{DA}^1$) in the spatial domain is shown at the top. The outlines in the middle of the bottom show the union of all neurons found in $\mathbf{D}$, superimposed onto a maximum intensity projection across the background-subtracted raw image sequence. The graphs on the right show a different view on the transients estimated for single neurons, that is, the rows of $\mathbf{U}^0$. The raw data comes from a mouse hippocampal slice, where single neurons can indeed be part of more than one assembly [20]. Analogous results on synthetic data are shown in the supplemental material.

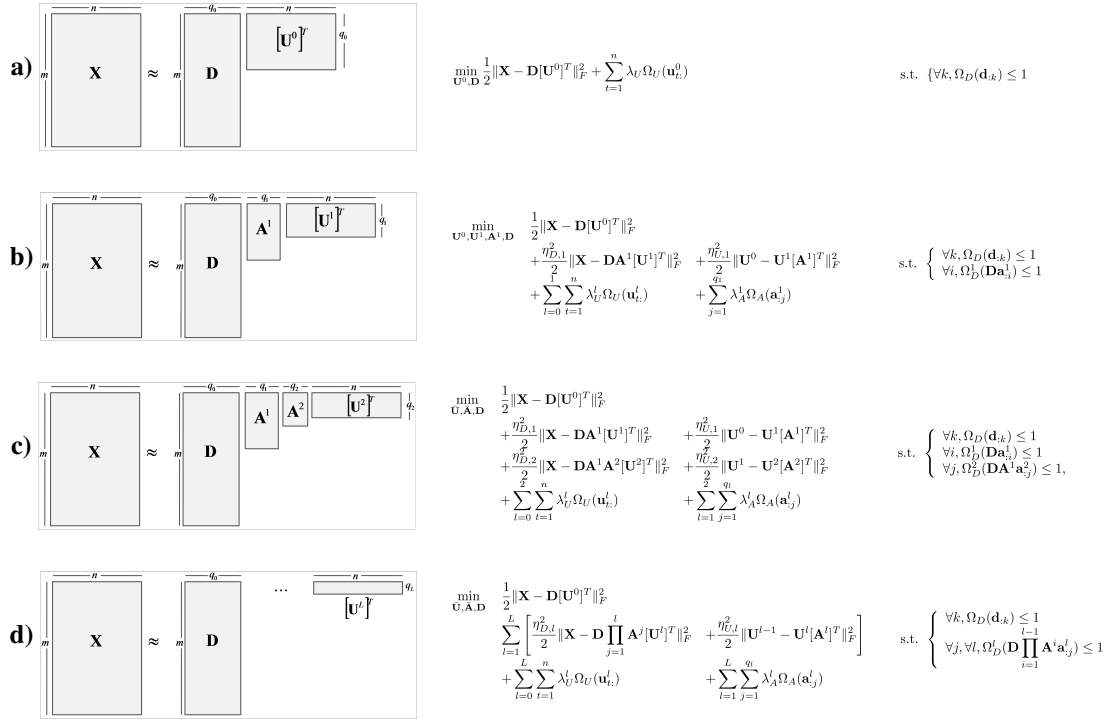

Figure 3: Decomposition of $\mathbf{X}$ into $\{1, 2, 3, L+1\}$ levels, with corresponding equations.

## 2.2 Bilevel Sparse Matrix Factorization

We now come to the heart of this work. To build intuition, we first refer to the application that has motivated this development, before giving mathematical details. The relation between the symbols used in the following is sketched in Fig. 3(b), while actual matrix contents are partially visualized in Fig. 2.

Given is a sequence of $n$ noisy sparse images which we vectorize and collect in the columns of matrix $\mathbf{X}$. We would like to find the following:

- a dictionary $\mathbf{D}$ of $q_0$ vectorized images comprising $m$ pixels each. Ideally, each basis function should correspond to a single neuron.
- a matrix $\mathbf{A}^1$ indicating to what extent each of the $q_0$ neurons is associated with any of the $q_1$ neuronal assemblies. We will call this matrix interchangeably *assignment* or *adjacency* matrix in the following. It is this matrix which encapsulates the quintessential structure we extract from the raw data, viz., which lower-level concept is associated with which higher-level concept.
- a coefficient matrix $[\mathbf{U}^1]^T$ that encodes in its rows the temporal evolution (activation) of the $q_1$ neuronal assemblies across $n$ time steps.
- a coefficient matrix $[\mathbf{U}^0]^T$ (shown in the equation, but not in the sketch of Fig. 3(b)) that encodes in its rows the temporal activation of the $q_0$ neuron basis functions across $n$ time steps.

The quantities $\mathbf{D}$, $\mathbf{A}^1$, $[\mathbf{U}^0]$, $[\mathbf{U}^1]$ in this redundant representation need to be consistent.

Let us now turn to equations. At first sight, it seems like minimizing $\|\mathbf{X} - \mathbf{D}\mathbf{A}^1[\mathbf{U}^1]^T\|_F^2$ over $\mathbf{D}, \mathbf{A}^1, \mathbf{U}^1$ subject to constraints should do the job. However, this could be too much of a simplification! To illustrate, assume for the moment that only a single neuronal assembly is active at any given time. Then all neurons associated with that assembly would follow an absolutely identical time course. While it is expected that neurons from an assembly show similar activation patterns [20], this is something we want to glean from the data, and not absolutely impose. In response, we introduce an auxiliary matrix $\mathbf{U}^0 \approx \mathbf{U}^1[\mathbf{A}^1]^T$ showing the temporal activation pattern of individual neurons. These two matrices, $\mathbf{U}^0$ and $\mathbf{U}^1$, are also shown in the false color plots of the collage of Fig. 2, bottom left.

The full equation involving coefficient and auxiliary coefficient matrices is shown in Fig. 3(b). The terms involving $\mathbf{X}$ are data fidelity terms, while $\|\mathbf{U}^0 - \mathbf{U}^1[\mathbf{A}^1]^T\|_F^2$ enforces consistency. Parameters $\eta$ trade off the various terms, and constraints of a different kind can be applied selectively to each of the matrices that we optimize over. Jointly optimizing over $\mathbf{D}, \mathbf{A}^1, \mathbf{U}^0$, and $\mathbf{U}^1$ is a hard and non-convex problem that we address using a block coordinate descent strategy described in section 2.4 and supplemental material.

## 2.3 Trilevel and Multi-level Sparse Matrix Factorization

We now discuss the generalization to an arbitrary number of levels that may be relevant for applications other than calcium imaging. To give a better feeling for the structure of the equations, the trilevel case is spelled out explicitly in Fig. 3(c), while Fig. 3(d) shows the general case of $L + 1$ levels.

The most interesting matrices, in many ways, are the assignment matrices $\mathbf{A}^1, \mathbf{A}^2$, etc. Assume, first, that the relations between lower-level and higher-level concepts obey a strict inclusion hierarchy. Such relations can be expressed in terms of a forest of trees: each highest-level concept is the root of a tree which fans out to all subordinate concepts. Each subordinate concept has a single parent only. Such a forest can also be seen as a (special case of an $L + 1$-partite) graph, with an adjacency matrix $\mathbf{A}^l$ specifying the parents of each concept at level $l - 1$. To impose an inclusion hierarchy, one can enforce the nestedness condition by requiring that $\|\mathbf{a}_{k:}^l\|_0 \leq 1$.

In general, and in the application considered here, one will not want to impose an inclusion hierarchy. In that case, the relations between concepts can be expressed in terms of a concatenation of bipartite graphs that conform with a directed acyclic graph. Again, the adjacency matrices encode the structure of such a directed acyclic graph.

In summary, the general equation in Fig. 3(d) is a principled alternative to simpler approaches that would impose the relations between concepts, or estimate them separately using, for instance, clustering algorithms; and that would then find a sparse factorization subject to this structure. Instead, we *simultaneously estimate the relation between concepts at different levels, as well as find a sparse approximation to the raw data*.

## 2.4 Optimization

The optimization problem in Fig. 3(d) is not *jointly* convex, but becomes convex w.r.t. one variable while keeping the others fixed provided that the norms $\Omega_U$, $\Omega_D$, and $\Omega_A$ are also convex. Indeed, it is possible to define convex norms that not only induce sparse solutions, but also favor non-zero patterns of a specific structure, such as sets of variables in a convex polygon with certain symmetry constraints [10]. Following [5], we use such norms to bias towards neuron basis functions holding a single neuron only. We employ a block coordinate descent strategy [2, Section 2.7] that iteratively optimizes one group of variables while fixing all others. Due to space limitations, the details and implementation of the optimization are described in the supplemental material.

# 3 Methods

## 3.1 Decomposition into neurons and their transients only

***Cell Sorting* [18] and *Adina* [5]**   focus only on the detection of cell centroids and of cell shape, and the estimation and analysis of Calcium transient signals. However, these methods provide no means to detect and identify neuronal co-activation. The key idea is to decompose calcium imaging data into constituent signal sources, *i.e.* temporal and spatial components. *Cell sorting* combines principal component analysis (PCA) and independent component analysis (ICA). In contrast, *Adina* relies on a matrix factorization based on sparse coding and dictionary learning [15], exploiting that neuronal activity is sparsely distributed in both space and time. Both methods are combined with a subsequent image segmentation since the spatial components (basis functions) often contain more than one neuron. Without such a segmentation step, overlapping cells or those with highly correlated activity are often associated with the same basis function.

## 3.2 Decomposition into neurons, their transients, and assemblies of neurons

***MNNMF+Adina***   Here, we combine a multilayer extension of non-negative matrix factorization with the segmentation from Adina. *MNNMF* [3] is a multi-stage procedure that iteratively decomposes the rightmost matrix of the decomposition that was previously found. In the first stage, we decompose the calcium imaging data into spatial and temporal components, just like the methods cited above, but using NMF and a non-negative least squares loss function [12] as implemented in [14]. We then use the segmentation from [5] to obtain single neurons in an updated dictionary[1] $\mathbf{D}$. Given this purged dictionary, the temporal components $\mathbf{U}^0$ are updated under the NMF criterion. Next, the temporal components $\mathbf{U}^0$ are further decomposed into two low-rank matrices, $\mathbf{U}^0 \approx \mathbf{U}^1[\mathbf{A}^1]^T$, again using NMF. Altogether, this procedure allows identifying neuronal assemblies and their temporal evolution. However, the *exact* number of assemblies $q_1$ must be defined a priori.

***KSVDS+Adina***   allows estimating a sparse decomposition [21] $\mathbf{X} \approx \mathbf{D}\mathbf{A}^1[\mathbf{U}^1]^T$ provided that $i)$ a dictionary of basis functions and $ii)$ the *exact* number of assemblies is supplied as input. In addition, the assignment matrix $\mathbf{A}^1$ is typically dense and needs to be thresholded. We obtain good results when supplying the purged dictionary[1] of single neurons resulting from Adina [5].

***SHMF – Sparse Heterarchical Matrix Factorization***   in its bilevel formulation decomposes the raw data simultaneously into neuron basis functions $\mathbf{D}$, a mapping of these to assemblies $\mathbf{A}^1$, as well as time courses of neurons $\mathbf{U}^0$ and assemblies $\mathbf{U}^1$, see equation in Fig. 3. Sparsity is induced by setting $\Omega_U$ and $\Omega_A$ to the $l_1$-norm. In addition, we impose the $l_2$-norm at the assembly level $\Omega_D^1$,

and let $\Omega_D$ be the structured sparsity-inducing norm proposed by Jenatton *et al.* [10]. In contrast to all other approaches described above, this already suffices to produce basis functions that contain, in most cases, only single neurons. Exceptions arise only in the case of cells which both overlap in space and have high temporal correlation. For this reason, and for a fair comparison with the other methods, we again use the segmentation from [5]. For the optimization, $\mathbf{D}$ and $\mathbf{U}^0$ are initialized with the results from Adina. $\mathbf{U}^1$ is initialized randomly with positive-truncated Gaussian noise, and $\mathbf{A}^1$ by the identity matrix as in KSVDS [21]. Finally, the number of neurons $q_0$ and neuronal assemblies $q_1$ are set to generous upper bounds of the expected true numbers, and are both set to equal values (here: $q_0 = q_1 = 60$) for simplicity. Note that a precise specification as for the above methods is not required.

# 4 Results

To obtain quantitative results, we first evaluate the proposed methods on synthetic image sequences designed so as to exhibit similar characteristics as the real data. We also report a qualitative analysis of the performance on real data from [20]. Since neuronal assemblies are still the subject of ongoing research, ground truth is not available for such real-world data.

## 4.1 Artifical Sequences

For evaluation, we created 80 synthetic sequences with 450 frames of size $128 \times 128$ pixels with a frame rate of $30 fps$. The data is created by randomly selecting cell shapes from 36 different active cells extracted from real data, and locating them in different locations with an overlap of up to $30\%$. Each cell is randomly assigned to up to three out of a total of five assemblies. Each assembly fires according to a dependent Poisson process, with transient shapes following a one-sided exponential decay with a scale of $500$ to $800ms$ that is convolved by a Gaussian kernel with $\sigma = 50ms$. The dependency is induced by eliminating all transients that overlap by more than $20\%$. Within such a transient, the neurons associated with the assembly fire with a probability of $90\%$ each. The number of cells per assembly varies from 1 to 10, and we use five assemblies in all experiments. Finally, the synthetic movies are distorted by white Gaussian noise with a relative amplitude, (max. intensity $-$ mean intensity)$/\sigma_{noise} \in \{3, 5, 7, 10, 12, 15, 17, 20\}$. By construction, the identity, location and activity patterns of all cells along with their membership in assemblies are known. The supplemental material shows one example, and two frames are shown in Fig. 1.

**Identificaton of assemblies** First, we want to quantify the ability to correctly infer assemblies from an image sequence. To that end, we compute the graph edit distance of the estimated assignments of neurons to assemblies, encoded in matrices $\mathbf{A}^1$, to the known ground truth. We count the number of false positive and false negative edges in the assignment graphs, where vertices (assemblies) are matched by minimizing the Hamming distance between binarized assignment matrices over all permutations.

Remember that MNNMF+Adina and KSVDS+Adina require a specification of the precise number of assemblies, which is unknown for real data. Accordingly, adjacency matrices, $\mathbf{A}^1 \in \mathbb{R}^{q_0 \times q_1}$ for different values for the number of assemblies $q_1 \in [3, 7]$ were estimated. Bilevel SHMF only needs an upper bound on the number of assemblies. Its performance is independent of the precise value, but computational cost increases with the bound. In these experiments, $q_1$ was set to 60.

Fig. 4 shows that all methods from section 3.2 give respectable performance in the task of inferring neuronal assemblies from nontrivial synthetic image sequences. For the true number of assemblies ($q_1 = 5$), Bilevel SHMF reaches a higher sensitivity than the alternative methods, with a median difference of 14%. According to the quartiles, the precisions achieved are broadly comparable, with MNNMF+Adina reaching the highest value.

All methods from section 3.2 also infer the temporal activity of all assemblies, $\mathbf{U}^1$. We omit a comparison of these matrices for lack of a good metric that would also take into account the correctness of the assemblies themselves: a fine time course has little worth if its associated assembly is deficient, for instance by having lost some neurons with respect to ground truth.

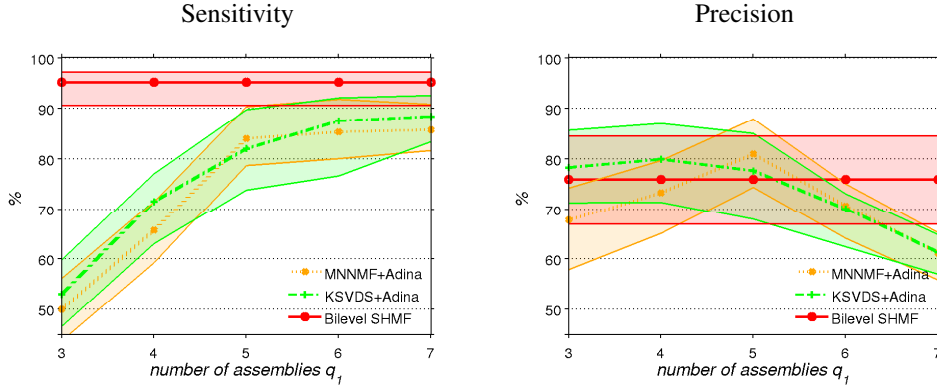

Figure 4: Performance on learning correct assignments of neurons to assemblies from nontrivial synthetic data with ground truth. KSVDS+Adina and MNNMF+Adina require that the number of assemblies $q_1$ be fixed in advance. In contrast, bilevel SHMF estimates the number of assemblies given an upper-bound. Its performance is hence shown as a constant over the $q_1$-axis. Plots show the median as well as the band between the lower and the upper quartile for all 80 sequences. Colors at non-integer $q_1$-values are a guide to the eye.

**Detection of calcium transients**  While the detection of assemblies as evaluated above is completely new in the literature, we now turn to a better studied [18, 5] problem: the detection of calcium transients of individual neurons. Some estimates for these characteristic waveforms are also shown, for real-world data, on the right hand side of Fig. 2.

To quantify transient detection performance, we compute the sensitivity and precision as in [20]. Here, sensitivity is the ratio of correctly detected to all neuronal activities; and precision is the ratio of correctly detected to all detected neuronal activities. Results are shown in Fig. 5.

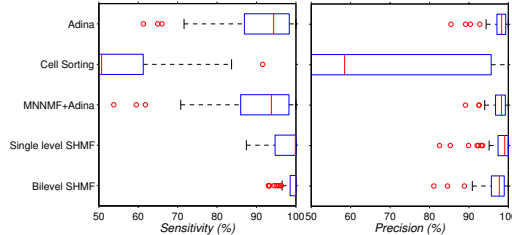

Figure 5: Sensitivity and precision of transient detection for individual neurons. Methods that estimate both assemblies and neuron transients perform at least as well as their simpler counterparts that focus on the latter.

Perhaps surprisingly, the methods from section 3.2 (MNNMF+Adina and Bilevel SHMF[2]) fare at least as well as those from section 3.1 (CellSorting and Adina). This is not self-evident, because a bilevel factorization could be expected to be more ill-posed than a single level factorization.

We make two observations: Firstly, it seems that using a bilevel representation with suitable regularization constraints helps stabilize the activity estimates also for single neurons. Secondly, the higher sensitivity and similar precision of bilevel SHMF compared to MNNMF+Adina suggest that a *joint* estimation of neurons, assemblies and their temporal activities as described in section 2 increases the robustness, and compensates errors that may not be corrected in greedy level-per-level estimation.

Incidentally, the great spread of both sensitivities and precisions results from the great variety of noise levels used in the simulations, and attests to the difficulty of part of the synthetic data sets.

| Raw data | Cell Sorting [18] | Adina [5] | Neurons $(\mathbf{D}[\mathbf{U}^0]^T)$ | Assemblies $(\mathbf{D}\mathbf{A}^1[\mathbf{U}^1]^T)$ |

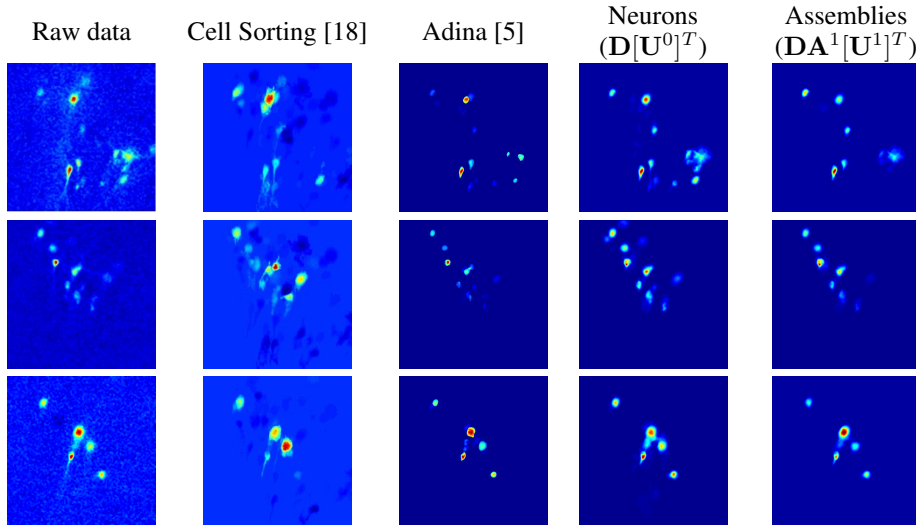

Figure 6: Three examples of raw data and reconstructed images of the times indicated in Fig. 2. The other examples are shown in the supplemental material.

## 4.2 Real Sequences

We have applied bilevel SHMF to epifluorescent data sets from mice (C57BL6) hippocampal slice cultures. As shown in Fig. 2, the method is able to distinguish overlapping cells and highly correlated cells, while at the same time estimating neuronal co-activation patterns (assemblies). Exploiting spatio-temporal sparsity and convex cell shape priors allows to accurately infer the transient events.

## 5  Discussion

The proposed multi-level sparse factorization essentially combines a clustering of concepts across several levels (expressed by the assignment matrices) with the finding of a basis dictionary, shared by concepts at all levels, and the finding of coefficient matrices for different levels. The formalism allows imposing different regularizers at different levels. Users need to choose tradeoff parameters $\eta, \lambda$ that indirectly determine the number of concepts (clusters) found at each level, and the sparsity. The ranks $q_l$, on the other hand, are less important: Figure 2 shows that the ranks of estimated matrices can be lower than their nominal dimensionality: superfluous degrees of freedom are simply not used.

On the application side, the proposed method allows to accomplish the detection of neurons, assemblies and their relation in a single framework, exploiting sparseness in the temporal and spatial domain in the process. Bilevel SHMF in particular is able to detect automatically, and differentiate between, overlapping and highly correlated cells, and to estimate the underlying co-activation patterns. As shown in Fig. 6, this approach is able to reconstruct the raw data at both levels of representations, and to make plausible proposals for neuron and assembly identification.

Given the experimental importance of calcium imaging, automated methods in the spirit of the one described here can be expected to become an essential tool for the investigation of complex activation patterns in live neural tissue.

## Acknowledgement

We are very grateful for partial financial support by CellNetworks Cluster (EXC81). We also thank Susanne Reichinnek, Martin Both and Andreas Draguhn for their comments on the manuscript.

## Footnotes

[1]Without such a segmentation step, the dictionary atoms often comprise more than one neuron, and overall results (not shown) are poor.

[2]KSVDS is not evaluated here because it does not yield activity estimates for individual neurons.

# References

[1] G. Bergqvist and E. G. Larsson. The Higher-Order Singular Value Decomposition Theory and an Application. *IEEE Signal Processing Magazine*, 27(3):151–154, 2010.

[2] D. P. Bertsekas. *Nonlinear Programming*. Athena Scientific, 1999.

[3] A. Cichocki and R. Zdunek. Multilayer nonnegative matrix factorization. *Electronics Letters*, 42:947–948, 2006.

[4] A. Cichocki, R. Zdunek, A. H. Phan, and S. Amari. *Nonnegative Matrix and Tensor Factorizations - Applications to Exploratory Multi-way Data Analysis and Blind Source Separation*. Wiley, 2009.

[5] F. Diego, S. Reichinnek, M. Both, and F. A. Hamprecht. Automated identification of neuronal activity from calcium imaging by sparse dictionary learning. In *International Symposium on Biomedical Imaging, in press*, 2013.

[6] W. Goebel and F. Helmchen. In vivo calcium imaging of neural network function. *Physiology*, 2007.

[7] C. Grienberger and A. Konnerth. Imaging calcium in neurons. *Neuron*, 2011.

[8] Q. Ho, J. Eisenstein, and E. P. Xing. Document hierarchies from text and links. In *Proc. of the 21st Int. World Wide Web Conference (WWW 2012)*, pages 739–748. ACM, 2012.

[9] R. Jenatton, A. Gramfort, V. Michel, G. Obozinski, E. Eger, F. Bach, and B. Thirion. Multi-scale Mining of fMRI data with Hierarchical Structured Sparsity. *SIAM Journal on Imaging Sciences*, 5(3), 2012.

[10] R. Jenatton, G. Obozinski, and F. Bach. Structured sparse principal component analysis. In *International Conference on Artificial Intelligence and Statistics (AISTATS)*, 2010.

[11] J. Kerr and W. Denk. Imaging in vivo: watching the brain in action. *Nature Review Neuroscience*, 2008.

[12] H. Kim and H. Park. Nonnegative matrix factorization based on alternating nonnegativity constrained least squares and active set method. *SIAM J. on Matrix Analysis and Applications*, 2008.

[13] S. Kim and E. P. Xing. Tree-guided group lasso for multi-response regression with structured sparsity, with an application to eQTL mapping. *Ann. Appl. Stat.*, 2012.

[14] Y. Li and A. Ngom. The non-negative matrix factorization toolbox for biological data mining. In *BMC Source Code for Biology and Medicine*, 2013.

[15] J. Mairal, F. Bach, J. Ponce, and G. Sapiro. Online dictionary learning for sparse coding. In *Proceedings of the 26th Annual International Conference on Machine Learning*, 2009.

[16] J. Mairal, F. Bach, J. Ponce, and G. Sapiro. Online Learning for Matrix Factorization and Sparse Coding. *Journal of Machine Learning Research*, 2010.

[17] J. Mairal, F. Bach, J. Ponce, G. Sapiro, and R. Jenatton. Sparse modeling software. http://spams-devel.gforge.inria.fr/.

[18] E. A. Mukamel, A. Nimmerjahn, and M. J. Schnitzer. Automated analysis of cellular signals from large-scale calcium imaging data. *Neuron*, 2009.

[19] M. Protter and M. Elad. Image sequence denoising via sparse and redundant representations. *IEEE Transactions on Image Processing*, 18(1), 2009.

[20] S. Reichinnek, A. von Kameke, A. M. Hagenston, E. Freitag, F. C. Roth, H. Bading, M. T. Hasan, A. Draguhn, and M. Both. Reliable optical detection of coherent neuronal activity in fast oscillating networks in vitro. *NeuroImage*, 60(1), 2012.

[21] R. Rubinstein, M. Zibulevsky, and M. Elad. Double sparsity: Learning sparse dictionaries for sparse signal approximation. *IEEE Transactions on Signal Processing*, 2010.

[22] A. P. Singh and G. J. Gordon. A unified view of matrix factorization models. *ECML PKDD*, 2008.

[23] M. Sun and H. Van Hamme. A two-layer non-negative matrix factorization model for vocabulary discovery. In *Symposium on machine learning in speech and language processing*, 2011.

[24] Q. Sun, P. Wu, Y. Wu, M. Guo, and J. Lu. Unsupervised multi-level non-negative matrix factorization model: Binary data case. *Journal of Information Security*, 2012.

[25] J. Yang, Z. Wang, Z. Lin, X. Shu, and T. S. Huang. Bilevel sparse coding for coupled feature spaces. In *CVPR'12*, pages 2360–2367. IEEE, 2012.

[26] B. Zhao, L. Fei-Fei, and E. P. Xing. Online detection of unusual events in videos via dynamic sparse coding. In *The Twenty-Fourth IEEE Conference on Computer Vision and Pattern Recognition*, Colorado Springs, CO, June 2011.

